# Pulling It All Together: Methods for Combining Neural Networks

Michael P. Perrone
Institute for Brain and Neural Systems
Brown University
Providence, RI
*mpp@cns.brown.edu*

The past several years have seen a tremendous growth in the complexity of the recognition, estimation and control tasks expected of neural networks. In solving these tasks, one is faced with a large variety of learning algorithms and a vast selection of possible network architectures. After all the training, how does one know which is the best network? This decision is further complicated by the fact that standard techniques can be severely limited by problems such as over-fitting, data sparsity and local optima. The usual solution to these problems is a winner-take-all cross-validatory model selection. However, recent experimental and theoretical work indicates that we can improve performance by considering methods for combining neural networks.

This workshop examined current neural network optimization methods based on combining estimates and task decomposition, including Boosting, Competing Experts, Ensemble Averaging, Metropolis algorithms, Stacked Generalization and Stacked Regression. The issues covered included Bayesian considerations, the role of complexity, the role of cross-validation, incorporation of a priori knowledge, error orthogonality, task decomposition, network selection techniques, over-fitting, data sparsity and local optima. Highlights of each talk are given below. To obtain the workshop proceedings, please contact the author or Norma Caccia (*norma_caccia@brown.edu*) and ask for IBNS ONR technical report #69.

M. Perrone (Brown University, "Averaging Methods: Theoretical Issues and Real World Examples") presented weighted averaging schemes [7], discussed their theoretical foundation [6], and showed that averaging can improve performance whenever the cost function is (positive or negative) convex which includes Mean Square Error, a general class of $L_p$-norm cost functions, Maximum Likelihood Estimation, Maximum Entropy, Maximum Mutual Information, the Kullback-Leibler Information (Cross Entropy), Penalized Maximum Likelihood Estimation and Smoothing Splines [6]. Averaging was shown to improve performance on the NIST OCR data, a human face recognition task and a time series prediction task [5].
J. Friedman (Stanford, "A New Approach to Multiple Outputs Using Stacking") presented a detailed analysis of a method for averaging estimators and noted simulations showed that averaging with a positivity constraint was better than cross-validation estimator selection [1].

S. Nowlan (Synaptics, "Competing Experts") emphasized the distinctions between static and dynamic algorithms and between averaged and stacked algorithms; and presented results of the mixture of experts algorithm [3] on a vowel recognition task and a hand tracking task.

H. Drucker (AT&T, "Boosting Compared to Other Ensemble Methods") reviewed the boosting algorithm [2] and showed how it can improve performance for OCR data.

J. Moody (OGI, "Predicting the U.S. Index of Industrial Production") showed that neural networks make better predictions for the US IP index than standard models [4] and that averaging these estimates improves prediction performance further.

W. Buntine (NASA Ames Research Center, "Averaging and Probabilistic Networks: Automating the Process") discussed placing combination techniques within the Bayesian framework.

D. Wolpert (Santa Fe Institute, "Inferring a Function vs. Inferring an Inference Algorithm") argued that theory can not, in general, identify the optimal network; so one must make assumptions in order to improve performance.

H. Thodberg (Danish Meat Research Institute, "Error Bars on Predictions from Deviations among Committee Members (within Bayesian Backprop)") raised the provocative (and contentious) point that Bayesian arguments support averaging while Occam's Razor (seemingly?) does not.

S. Hashem (Purdue University, "Merits of Combining Neural Networks: Potential Benefits and Risks") emphasized the importance of dealing with collinearity when using averaging methods.

# References

[1] Leo Breiman. Stacked regression. Technical Report TR-367, Department of Statistics, University of California, Berkeley, August 1992.

[2] Harris Drucker, Robert Schapire, and Patrice Simard. Boosting performance in neural networks. *International Journal of Pattern Recognition and Artificial Intelligence*, [To appear].

[3] R. A. Jacobs, M. I. Jordan, S. J. Nowlan, and G. E. Hinton. Adaptive mixtures of local experts. *Neural Computation*, 3(2), 1991.

[4] U. Levin, T. Leen, and J. Moody. Fast pruning using principal components. In Steven J. Hanson, Jack D. Cowan, and C. Lee Giles, editors, *Advances in Neural Information Processing Systems 6*. Morgan Kaufmann, 1994.

[5] M. P. Perrone. *Improving Regression Estimation: Averaging Methods for Variance Reduction with Extensions to General Convex Measure Optimization*. PhD thesis, Brown University, Institute for Brain and Neural Systems; Dr. Leon N Cooper, Thesis Supervisor, May 1993.

[6] M. P. Perrone. General averaging results for convex optimization. In *Proceedings of the 1993 Connectionist Models Summer School*, pages 364–371, Hillsdale, NJ, 1994. Erlbaum Associates.

[7] M. P. Perrone and L. N Cooper. When networks disagree: Ensemble method for neural networks. In *Artificial Neural Networks for Speech and Vision*. Chapman-Hall, 1993. Chapter 10.

